# Iterative Double Clustering for Unsupervised and Semi-Supervised Learning

**Ran El-Yaniv**    **Oren Souroujon**
Computer Science Department
Technion - Israel Institute of Technology
(rani,orenso)@cs.technion.ac.il

## Abstract

We present a powerful meta-clustering technique called Iterative Double Clustering (IDC). The IDC method is a natural extension of the recent Double Clustering (DC) method of Slonim and Tishby that exhibited impressive performance on text categorization tasks [12]. Using synthetically generated data we empirically find that whenever the DC procedure is successful in recovering some of the structure hidden in the data, the extended IDC procedure can incrementally compute a significantly more accurate classification. IDC is especially advantageous when the data exhibits high attribute noise. Our simulation results also show the effectiveness of IDC in text categorization problems. Surprisingly, this unsupervised procedure can be competitive with a (supervised) SVM trained with a small training set. Finally, we propose a simple and natural extension of IDC for semi-supervised and transductive learning where we are given both labeled and unlabeled examples.

## 1    Introduction

Data clustering is a fundamental and challenging routine in information processing and pattern recognition. Informally, when we cluster a set of elements we attempt to partition it into subsets such that points in the same subset are more "similar" to each other than to points in other subsets. Typical clustering algorithms depend on a choice of a similarity measure between data points [6], and a "correct" clustering result depends on an appropriate choice of a similarity measure. However, the choice of a "correct" measure is an ill-defined task without a particular application at hand. For instance, consider a hypothetical data set containing articles by each of two authors such that half of the articles authored by each author discusses one topic, and the other half discusses another topic. There are two possible dichotomies of the data which could yield two different bi-partitions: one according to topic, and another, according to writing style. When asked to cluster this set into two sub-clusters, one cannot successfully achieve the task without knowing the goal: Are we interested in clusters that reflect writing style or semantics? Therefore, without a suitable target at

hand and a principled method for choosing a similarity measure suitable for the target, it can be meaningless to interpret clustering results.

The *information bottleneck (IB)* method of Tishby, Pereira and Bialek [8] is a recent framework that can sometimes provide an elegant solution to this problematic "metric selection" aspect of data clustering (see Section 2). The original IB method generates a soft clustering assignments for the data. In [10], Slonim and Tishby developed a simplified "hard" variant of the IB clustering, where there is a hard assignment of points to their clusters. Employing this hard IB clustering, the same authors introduced an effective two-stage clustering procedure called *Double Clustering (DC)* [12]. An experimental study of DC on text categorization tasks [12] showed a consistent advantage of DC over other clustering methods. A striking finding in [12] is that DC sometimes even attained results close to those of supervised learning.[1]

In this paper we present a powerful extension of the DC procedure which we term *Iterative Double Clustering (IDC)*. IDC performs iterations of DC and whenever the first DC iteration succeeds in extracting a meaningful structure of the data, a number of the next consecutive iterations can continually improve the clustering quality. This continual improvement achieved by IDC is due to generation of progressively less noisy data representations which reduce variance. Using synthetically generated data, we study some properties of IDC. Not only that IDC can dramatically outperform DC whenever the data is noisy, our experiments indicate that IDC attains impressive categorization results on text categorization tasks. In particular, we show that our unsupervised IDC procedure is competitive with an SVM (and Naive Bayes) trained over a small sized training set. We also propose a natural extension of IDC for transductive semi-supervised transductive. Our preliminary empirical results indicate that our transductive IDC can yield effective text categorization.

## 2    Information Bottleneck and Double Clustering

We consider a data set $X$ of *elements*, each of which is a $d$-dimensional vector over a set $F$ of *features*. We focus on the case where feature values are non-negative real numbers. For every element $x = (f_1, \ldots, f_d) \in X$ we consider the empirical conditional distribution $\{p(f_i|x)\}$ of features given $x$, where $p(f_i|x) = f_i / \sum_{i=1}^{d} f_i$. For instance, $X$ can be a set of documents, each of which is represented as a vector of word-features where $f_i$ is the frequency of the $i$th word (in some fixed word enumeration). Thus, we represent each element as a distribution over its features, and are interested in a partition of the data based on these feature conditional distributions. Given a predetermined number of clusters, a straightforward approach to cluster the data using the above "distributional representation" would be to choose some (dis)similarity measure for distributions (e.g. based on some $L_p$ norm or some statistical measure such as the KL-divergence) and employ some "plug-in" clustering algorithm based on this measure (e.g. agglomerative algorithms). Perhaps due to feature noise, this simplistic approach can result in mediocre results (see e.g. [12]).

Suppose that our data is given via observations of a random variable $S$. In the *information bottleneck (IB)* method of Tishby *et al.* [8] we attempt to extract the essence of the data $S$ using co-occurrence observations of $S$ together with a target variable $T$. The goal is to extract a compressed representation $\tilde{S}$ of $S$ with minimum compromise of information content with respect to $T$. This way, $T$ can direct us to extract meaningful clustering from $S$ where the meaning is determined by the target $T$. Let $I(S,T) = \sum_{s \in S, t \in T} p(s,t) \log \frac{p(s,t)}{p(s)p(t)}$, the *mutual information* between $S$ and $T$ [3].

The IB method attempts to compute $p(\tilde{s}|s)$, a "soft" assignment of a data point $s$ to clusters $\tilde{s}$, so as to minimize $I(S, \tilde{S}) - \beta I(\tilde{S}, T)$, given the Markov condition $T \to S \to \tilde{S}$ (i.e., $T$ and $\tilde{S}$ are conditionally independent given $S$). Here, $\beta$ is a Lagrange multiplier that controls a constraint on $I(\tilde{S}, T)$ and thus, the tradeoff between the desired compression level and the predictive power of $\tilde{S}$ with respect to $T$. As shown in [8], this minimization yields a system of coupled equations for the clustering mapping $p(\tilde{s}|s)$ in terms of the cluster representations $p(t|\tilde{s})$ and the cluster weights $p(\tilde{s})$. The paper [8] also presents an algorithm similar to deterministic annealing [9] for recovering a solution for the coupled equations.

Slonim and Tishby [10] proposed a simplified IB approach for the computation of "hard" cluster assignments. In this hard IB variant, each data point $s$, represented by $\{p(t|s)\}_t$, is associated with one centroid $\tilde{s}$. They also devised a greedy agglomerative clustering algorithm that starts with the trivial clustering, where each data point $s$ is a single cluster; then, at each step, the algorithm merges the two clusters that minimize the loss of mutual information $I(\tilde{S}, T)$. The reduction in $I(\tilde{S}, T)$ due to a merge of two clusters $\tilde{s}_i$ and $\tilde{s}_j$ is shown to be

$$(p(\tilde{s}_i) + p(\tilde{s}_j))D_{JS}[p(t|\tilde{s}_i), p(t|\tilde{s}_j)], \qquad (1)$$

where, for any two distributions $p(x)$ and $q(x)$, with priors $\lambda_p$ and $\lambda_q$, $\lambda_p + \lambda_q = 1$, $D_{JS}[p(x), q(x)]$ is the *Jensen-Shannon divergence* (see [7, 4]),

$$D_{JS}[p(x), q(x)] = \lambda_p D_{KL}(p||\frac{p+q}{2}) + \lambda_q D_{KL}(q||\frac{p+q}{2}).$$

Here, $\frac{p+q}{2}$ denotes the distribution $(p(x)+q(x))/2$ and $D_{KL}(\cdot||\cdot)$ is the Kullbak-Leibler divergence [3]. This agglomerative algorithm is of course only *locally* optimal, since at each step it greedily merges the two most similar clusters. Another disadvantage of this algorithm is its time complexity of $O(n^2)$ for a data set of $n$ elements (see [12] for details).

The IB method can be viewed as a *meta-clustering* procedure that, given observations of the variables $S$ and $T$ (via their empirical co-occurrence samples $p(s, t)$), attempts to cluster $s$-elements represented as distributions over $t$-elements. Using the merging cost of equation (1) one can approximate IB clustering based on other "plug-in" vectorial clustering routines applied within the simplex containing the $s$-elements distributional representations.

DC [12] is a two-stage procedure where during the first stage we IB-cluster features represented as distributions over elements, thus generating *feature clusters*. During the second stage we IB-cluster elements represented as distributions over the feature *clusters* (a more formal description follows). For instance, considering a document clustering domain, in the first stage we cluster words as distributions over documents to obtain word clusters. Then in the second stage we cluster documents as distributions over word clusters, to obtain document clusters.

Intuitively, the first stage in DC generates more coarse *pseudo* features (i.e. feature centroids), which can reduce noise and sparseness that might be exhibited in the original feature values. Then, in the second stage, elements are clustered as distributions over the "distilled" pseudo features, and therefore can generate more accurate element clusters. As reported in [12], this DC two-stage procedure outperforms various other clustering approaches as well as DC variants applied with other dissimilarity measures (such as the variational distance) different from the optimal JS-divergence of Equation (1). It is most striking that in some cases, the accuracy achieved by DC was close to that achieved by a supervised Naive Bayes classifier.

# 3 Iterative Double Clustering (IDC)

Denote by $IB_N(T|S)$ the clustering result, into $N$ clusters, of the IB hard clustering procedure when the data is $S$ and the target variable is $T$ (see Section 2). For instance, if $T$ represents documents and $S$ represents words, the application of $IB_N(T = \text{documents}|S = \text{words})$ will cluster the words, represented as distributions over the documents, into $N$ clusters. Using the notation of our problem setup, with $X$ denoting the data and $F$ denoting the features, Figure 1 provides a pseudo-code of the IDC meta-clustering algorithm, which clusters $X$ into $N_{\tilde{X}}$ clusters. Note that the DC procedure is simply an application of IDC with $k = 1$.

The code of Figure 1 requires to specify $k$, the number of IDC iterations to run, $N_{\tilde{X}}$, the number of element clusters (e.g. the desired number of of document clusters) and $N_{\tilde{F}}$, the number of feature clusters to use during each iteration. In the experiments reported below we always assumed that we know the correct $N_{\tilde{X}}$. Our experiments show that the algorithm is not too sensitive to an overestimate of $N_{\tilde{F}}$. Note that the choice of these parameters is the usual model order selection problem. Perhaps the first question regarding $k$ (number of iterations) to ask is whether or not IDC *converges* to a steady state (e.g. where two consecutive iterations generate identical partitions). Unfortunately, a theoretical understanding of this convergence issue is left open in this paper. In most of our experiments IDC converged after a small number of iterations. In all the experiments reported below we used a fixed $k = 15$.

The "hard" IB-clustering originally presented by [12] uses an agglomerative procedure as its underlying clustering algorithm (see Section 2). The "soft" IB [8] applies a deterministic annealing clustering [9] as its underlying procedure. As already discussed, the IB method can be viewed as *meta-clustering* which can employ many vectorial clustering routines. We implemented IDC using several routines including agglomerative clustering and deterministic annealing. Since both these algorithms are computationally intensive, we also implemented IDC using a simple fast algorithm called *Add-C* proposed by Guedalia et al. [5]. Add-C is an online greedy clustering algorithm with linear

```
Input:
X (input data)
N_X̃ (number of element clusters)
N_F̃ (number of feature clusters to use)
k (number of iterations)
Initialize: S ← F, T ← X,
loop {k times}
    N ← N_F̃
    F̃ ← IB_N(T|S)
    N ← N_X̃, S ← X, T ← F̃
    X̃ ← IB_N(T|S)
    S ← F, T ← X̃
end loop
Output X̃
```

Figure 1: Pseudo-code for IDC

running time and can be viewed as a simple online approximation of $k$-means. For this reason, all the results reported below were computed using Add-C (whose description is omitted, for lack of space, see [5] for details). For obtaining a better approximation to the IB method we of course used the JS-divergence of (1) as our cost measure.

Following [12] we chose to evaluate the performance of IDC with respect to a *labeled* data set. Specifically, we count the number of classification errors made by IDC as obtained from labeled data.

In order to better understand the properties of IDC, we first examined it within a controlled setup of synthetically generated data points whose feature values were generated by $d$-dimensional Gaussian distributions (for $d$ features) of the form $N(\mu, \Sigma)$, where $\Sigma = \sigma^2 I$, with $\sigma$ constant. In order to simulate different sources, we assigned different $\mu$ values (from a given constant range) to each combination of source and feature. Specifically, for data simulating $m$ classes and $|F|$ features, $|F| \times m$ differ-

ent distributions were selected. We introduced feature noise by distorting each entry with value $v$ by adding a random sample from $N(0, (\alpha \cdot v)^2)$, where $\alpha$ is the "noise amplitude" (resulting negative values were rounded to zero). In figure 2(a), we plot the average accuracy of 10 runs of IDC. As can be seen, at low level noise amplitudes IDC attains perfect accuracy. When the noise amplitude increases, both IDC and DC deteriorate but the multiple rounds of IDC can better resist the extra noise. After observing the large accuracy gain between DC and IDC at a specific interval of noise amplitude within the feature noise setup, we set the noise amplitude to values in that interval and examined the behavior of the IDC run in more detail. Figure 2(b) shows a typical trace of the accuracy obtained at each of the 20 iterations of an IDC run over noisy data. This learning curve shows a quick improvement in accuracy during the first few rounds, and then reaches a plateau.

Following [12] we used the *20 Newsgroups (NG20)* [1] data set to evaluate IDC on real, labeled data. We chose several subsets of NG20 with various degrees of difficulty. In the first set of experiments we used the following four newsgroups (denoted as NG4), two of which deal with sports subjects: 'rec.sport.baseball', 'rec.sport.hockey', 'alt.atheism' and 'sci.med'. In these experiments we tested some basic properties of IDC. In all the experiments reported in this section we performed the following preprocessing: We lowered the case of all letters, filtered out low frequency words which appeared up to (and including) 3 times in the entire set and filtered out numerical and non-alphabetical characters. Of course we also stripped off newsgroup headers which contain the class labels.

In Figure 2(c) we display accuracy vs. number of feature clusters ($N_{\tilde{F}}$). The accuracy deteriorates when $N_{\tilde{F}}$ is too small and we see a slight negative trend when it increases. We performed an additional experiment which tested the performance using very large numbers of feature clusters. Indeed, these results indicate that after a plateau in the range of 10-20 there is a minor negative trend in the accuracy level. Thus, with respect to this data set, the IDC algorithm is not too sensitive to an overestimation of the number $N_{\tilde{F}}$ of feature clusters.

Other experiments over the NG4 data set confirmed the results of [12] that the JS-divergence dissimilarity measure of Equation (1) outperforms other measures, such as the variational distance ($L_1$ norm), the KL-divergence, the square-Euclidean distance and the 'cosine' distance. Details of all these experiments will be presented in the full version of the paper.

In the next set of experiments we tested IDC's performance on the same newsgroup subsets used in [12]. Table 1(a) compares the accuracy achieved by DC to the the last (15th) round of IDC with respect to all data sets described in [12]. Results of DC were taken from [12] where DC is implemented using the agglomerative routine.

Table 1(b) displays a preliminary comparison of IDC with the results of a Naive Bayes (NB) classifier (reported in [11]) and a support vector machine (SVM). In each of the 5 experiments the supervised classifiers were trained using 25 documents per class and tested on 475 documents per class. The input for the unsupervised IDC was 500 unlabeled documents per class. As can be seen, IDC outperforms in this setting both the naive Bayes learner and the SVM.

## 4  Learning from Labeled and Unlabeled Examples

In this section, we present a natural extension of IDC for semi-supervised *transductive* learning that can utilize *both* labeled and unlabeled data. In transductive learning, the testing is done on the unlabeled examples in the training data, while in semi-supervised

| Newsgroup | DC | IDC-15 |
|---|---|---|
| $Binary1$ | 0.70 | 0.85 |
| $Binary2$ | 0.68 | 0.83 |
| $Binary3$ | 0.75 | 0.80 |
| $Multi5_1$ | 0.59 | 0.86 |
| $Multi5_2$ | 0.58 | 0.88 |
| $Multi5_3$ | 0.53 | 0.86 |
| $Multi10_1$ | 0.35 | 0.56 |
| $Multi10_2$ | 0.35 | 0.49 |
| $Multi10_3$ | 0.35 | 0.55 |
| **Average** | **0.54** | **0.74** |

| Data Set | NB | SVM | IDC-15 | IDC-1 |
|---|---|---|---|---|
| COMP (5) | 0.50 | 0.51 | 0.50 | 0.34 |
| SCIENCE (4) | 0.73 | 0.68 | 0.79 | 0.44 |
| POLITICS (3) | 0.67 | 0.76 | 0.78 | 0.42 |
| RELIGION (3) | 0.55 | 0.78 | 0.60 | 0.38 |
| SPORT (2) | 0.75 | 0.78 | 0.89 | 0.76 |
| **Average** | **0.64** | **0.70** | **0.71** | **0.47** |

Table 1: **Left:** Accuracy of DC vs. IDC on most of the data sets described in [12]. DC results are taken from [12]; **Right:** Accuracy of Naive Bayes (NB) and SVM classifiers vs. IDC on some of the data sets described in [11]. The IDC-15 column shows final accuracy achieved at iteration 15 of IDC; the IDC-1 column shows first iteration accuracy. The NB results are taken from [11]. The SVM results were produced using the LibSVM package [2] with its default parameters. In all cases the SVM was trained and tested using the same training/test set sizes as described in [11] (25 documents per newsgroup for training and 475 for testing; the number of unlabeled documents fed to IDC was 500 per newsgroup). The number of newsgroups in each hyper-category is specified in parenthesis (e.g. COMP contains 5 newsgroups).

inductive learning it is done on previously unseen data. Here we only deal with the transductive case. In the full version of the paper we will present a semi-supervised inductive learning version of IDC.

For motivating the transductive IDC, consider a data set $X$ that has emerged from a statistical mixture which includes several sources (classes). Let $C$ be a random variable indicating the class of a random point. During the first iteration of a standard IDC we cluster the features $F$ so as to preserve $I(F, X)$. Typically, $X$ contains predictive information about the classes $C$. In cases where $I(X, C)$ is sufficiently large, we expect that the feature clusters $\tilde{F}$ will preserve some information about $C$ as well. Having available some *labeled* data points, we may attempt to generate feature clusters $\tilde{F}$ which preserve more information about class labels. This leads to the following straightforward idea. During the first IB-stage of the IDC first iteration, we cluster the features $F$ as distributions over *class labels* (given by the labeled data). This phase results in feature clusters $\tilde{F}$. Then we continue as usual; that is, in the second IB-phase of the first IDC iteration we cluster $X$, represented as distributions over $\tilde{F}$. Subsequent IDC iterations use all the unlabeled data.

In Figure 2(d) we show the accuracy obtained by DC and IDC in categorizing 5 newsgroups as a function of the training (labeled) set size. For instance, we see that when the algorithm has 10 documents available from each class it can categorize the entire unlabeled set, containing 90 unlabeled documents in each of the classes, with accuracy of about 80%. The benchmark accuracy of IDC with no labeled examples obtained about 73%.

In Figure 2(e) we see the accuracy obtained by DC and transductive IDC trained with a constant set of 50 labeled documents, on different unlabeled (test) sample sizes. The graph shows that the accuracy of DC significantly degrades, while IDC manages to sustain an almost constant high accuracy.

# 5 Concluding Remarks

Our contribution is threefold. First, we present a natural extension of the successful double clustering algorithm of [12]. Empirical evidence indicates that our new iterative DC algorithm has distinct advantages over DC, especially in noisy settings. Second, we applied the *unsupervised* IDC on text categorization problems which are typically dealt with by supervised learning algorithms. Our results indicate that it is possible to achieve performance competitive to supervised classifiers that were trained over small samples. Finally, we present a natural extension of IDC that allows for transductive learning. Our preliminary empirical evaluation of this scheme over text categorization appears to be promising.

A number of interesting questions are left for future research. First, it would be of interest to gain better theoretical understanding of several issues: the generalization properties of DC and IDC, the convergence of IDC to a steady state and precise conditions on attribute noise settings within which IDC is advantageous. Second, it would be important to test the empirical performance of IDC with respect to different problem domains. Finally, we believe it would be of great interest to better understand and characterize the performance of transductive IDC in settings having both labeled and unlabeled data.

## Acknowledgements

We thank Naftali Tishby and Noam Slonim for helpful discussions and for providing us with the detailed descriptions of the NG20 data sets used in their experiments. We also thank Ron Meir, Yiftach Ravid and the anonymous referees for their constructive comments. This research was supported by the Israeli Ministry of Science

## Footnotes

[1]Specifically, the DC method obtained in some cases accuracy close to that obtained by a naive Bayes classifier trained over a small sized sample [12].

## References

[1] 20 newsgroup data set. http://www.ai.mit.edu/~jrennie/20_newsgroups/.

[2] Libsvm. http://www.csie.ntu.edu.tw/~cjlin/libsvm.

[3] T.M. Cover and J.A. Thomas. *Elements of Information Theory.* John Wiley & Sons, Inc., 1991.

[4] R. El-Yaniv, S. Fine, and N. Tishby. Agnostic classification of markovian sequences. In *NIPS97*, 1997.

[5] I.D. Guedalia, M. London, and M. Werman. A method for on-line clustering of non-stationary data. *Neural Computation*, 11:521–540, 1999.

[6] A.K. Jain and R.C. Dubes. *Algorithms for Clustering Data.* Prentice-Hall, New Jersey, 1988.

[7] J. Lin. Divergence measures based on the shannon entropy. *IEEE Transactions on Information Theory*, 37(1):145–151, 1991.

[8] F.C. Pereira N. Tishby and W. Bialek. Information bottleneck method. In *37-th Allerton Conference on Communication and Computation*, 1999.

[9] K. Rose. Deterministic annealing for clustering, compression, classification, regression and related optimization problems. *Proceedings of the IEEE*, 86(11):2210–2238, 1998.

[10] N. Slonim and N. Tishby. Agglomerative information bottleneck. In *NIPS99*, 1999.

[11] N. Slonim and N. Tishby. The power of word clustering for text classification. To appear in the European Colloquium on IR Research, ECIR, 2001.

[12] Noam Slonim and Naftali Tishby. Document clustering using word clusters via the information bottleneck method. In *ACM SIGIR 2000*, 2000.

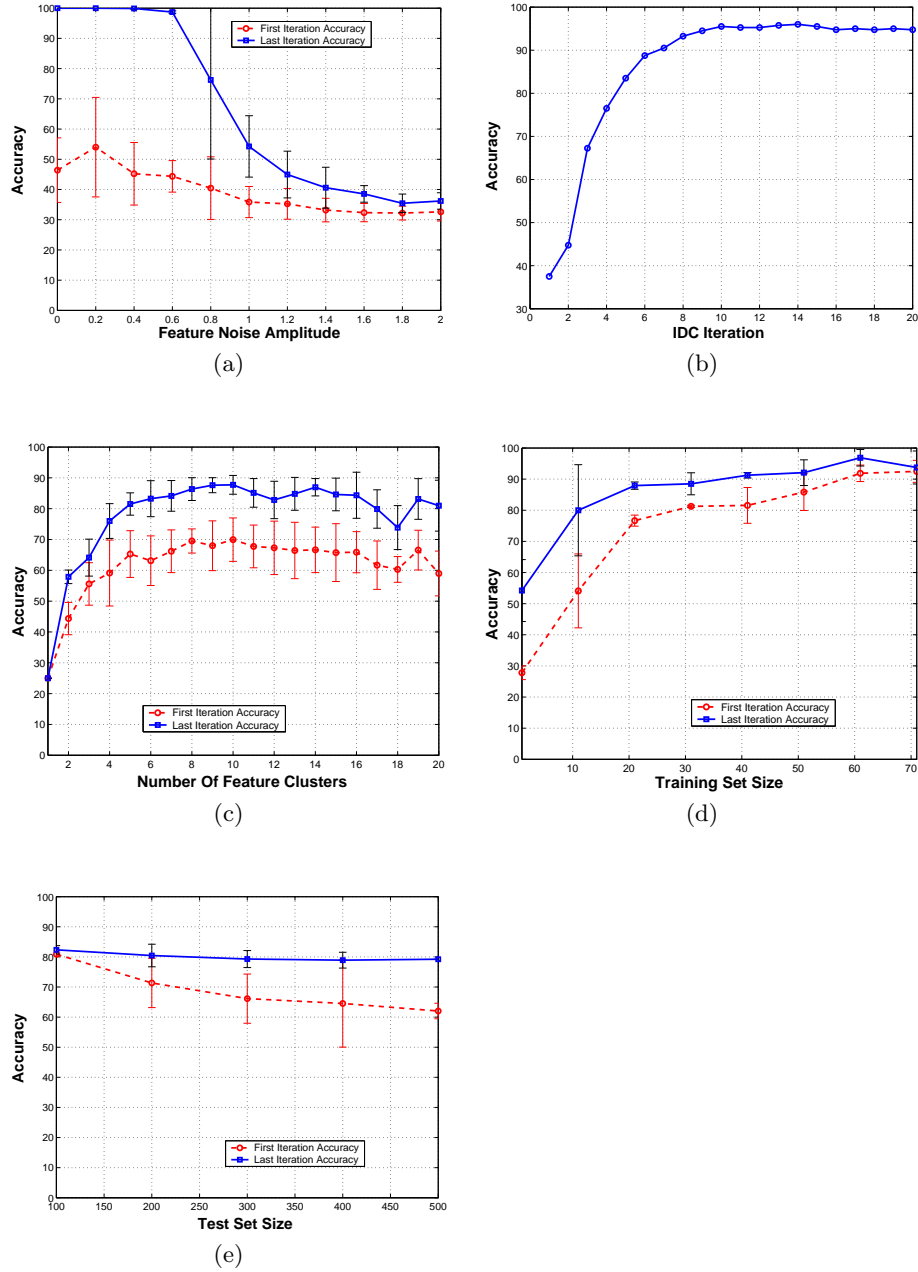

Figure 2: (a) Average accuracy over 10 trials for different amplitudes of proportional feature noise. Data set: A synthetically generated sample of 200 500-dimensional elements in 4 classes. (b) A trace of a single IDC run. The $x$-axis is the number of IDC iterations and the $y$-axis is accuracy achieved in each iteration. Data set: Synthetically generated sample of 500, 400-dimensional elements in 5 classes; Noise: Proportional feature noise with $\alpha = 1.0$; (c) Average accuracy (10 trials) for different numbers of feature clusters. Data set: NG4. (d) Average accuracy of (10 trials of) transductive categorization of 5 newsgroups. Sample size: 80 documents per class, X-axis is training set size. Upper curve shows trans. IDC-15 and lower curve is trans. IDC-1. (e) Average accuracy of (10 trials of) transductive categorization of 5 newsgroups. Sample size: constant training set size of 50 documents from each class. The $x$-axis counts the number of unlabeled samples to be categorized. Upper curve is trans. IDC-15 and lower curve is trans. IDC-1. Each error bar (in all graphs) specifies one std.